# Figure of Merit Training for Detection and Spotting

**Eric I. Chang and Richard P. Lippmann**
MIT Lincoln Laboratory
Lexington, MA 02173-0073, USA

## Abstract

Spotting tasks require detection of target patterns from a background of richly varied non-target inputs. The performance measure of interest for these tasks, called the *figure of merit* (FOM), is the detection rate for target patterns when the false alarm rate is in an acceptable range. A new approach to training spotters is presented which computes the FOM gradient for each input pattern and then directly maximizes the FOM using backpropagation. This eliminates the need for thresholds during training. It also uses network resources to model Bayesian *a posteriori* probability functions accurately only for patterns which have a significant effect on the detection accuracy over the false alarm rate of interest. FOM training increased detection accuracy by 5 percentage points for a hybrid radial basis function (RBF) – hidden Markov model (HMM) wordspotter on the credit-card speech corpus.

## 1 INTRODUCTION

Spotting tasks require accurate detection of target patterns from a background of richly varied non-target inputs. Examples include keyword spotting from continuous acoustic input, spotting cars in satellite images, detecting faults in complex systems over a wide range of operating conditions, detecting earthquakes from continuous seismic signals, and finding printed text on images which contain complex graphics. These problems share three common characteristics. First, the number of instances of target patterns is unknown. Second, patterns from background, non-target, classes are varied and often difficult to model accurately. Third, the performance measure of interest, called the *figure of merit* (FOM), is the detection rate for target patterns when the false alarm rate is over a specified range.

Neural network classifiers are often used for detection problems by training on target and background classes, optionally normalizing target outputs using the background output,

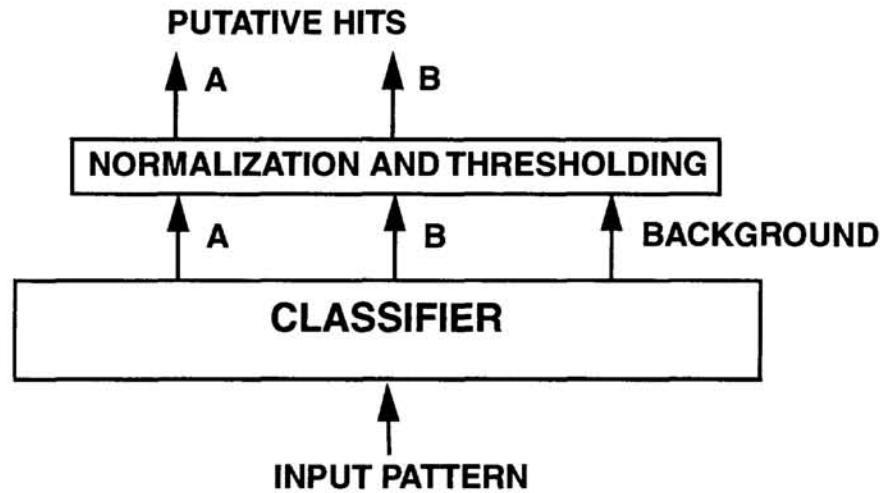

Figure 1. Block diagram of a spotting system.

and thresholding the resulting score to generate putative hits, as shown in Figure 1. Putative hits in this figure are input patterns which generate normalized scores above a threshold. We have developed a hybrid radial basis function (RBF) – hidden Markov model (HMM) keyword spotter. This wordspotter was evaluated using the NIST credit card speech database as in (Rohlicek, 1993, Zeppenfeld, 1993) using the same train/evaluation split of the training conversations as was used in (Zeppenfeld, 1993). The system spots 20 target keywords, includes one general filler class, and uses a Viterbi decoding backtrace as described in (Lippmann, 1993) to backpropagate errors over a sequence of input speech frames. The performance of this spotting system and its improved versions is analyzed by plotting detection versus false alarm rate curves as shown in Figure 2. These curves are generated by adjusting the classifier output threshold to allow few or many putative hits. Wordspotter putative hits used to generate Figure 2 correspond to speech frames when the difference between the cumulative log Viterbi scores in output HMM nodes of word and filler models is above a threshold. The FOM for this wordspotter is defined as the average keyword detection rate when the false alarm rate ranges from 1 to 10 false alarms per keyword per hour. The 69.7% figure of merit for this system means that 69.7% of keyword occurrences are detected on the average while generating from 20 to 200 false alarms per hour of input speech.

## 2  PROBLEMS WITH BACKPROPAGATION TRAINING

Neural network classifiers used for spotting tasks can be trained using conventional backpropagation procedures with *1 of N* desired outputs and a squared error cost function. This approach to training does not maximize the FOM because it attempts to estimate Bayesian *a posteriori* probability functions accurately for all inputs even if a particular input has little effect on detection accuracy at false alarm rates of interest. Excessive network resources may be allocated to modeling the distribution of common background inputs dissimilar from targets and of high-scoring target inputs which are easily detected. This problem can be addressed by training only when network outputs are above thresholds. This approach is problematic because it is difficult to set the threshold for different keywords, because using fixed target values of 1.0 and 0.0 requires careful normalization of network output scores to prevent saturation and maintain backpropagation effectiveness, and because the gradient calculated from a fixed target value does not reflect the actual impact on the FOM.

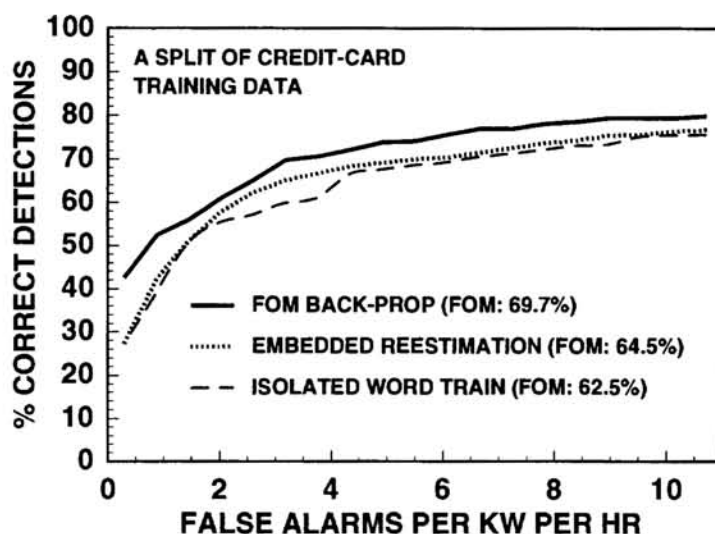

Figure 2. Detection vs. false alarm rate curve for a 20-word hybrid wordspotter.

Figure 3 shows the gradient of true hits and false alarms when target values are set to be 1.0 for true hits and 0.0 for false alarms, the output unit is sigmoidal, and the threshold for a putative hit is set to roughly 0.6. The gradient is the derivative of the squared error cost with respect to the input of the sigmodal output unit. As can be seen, low-scoring hits or false alarms that may affect the FOM are ignored, the gradient is discontinuous at the threshold, the gradient does not fall to zero fast enough at high values, and the relative sizes of the hit and false alarm gradients do not reflect the true effect of a hit or false alarm on the FOM.

## 3  FIGURE OF MERIT TRAINING

A new approach to training a spotter system called "figure of merit training" is to directly compute the FOM and its derivative. This derivative is the change in FOM over the change in the output score of a putative hit and can be used instead of the derivative of a squared-error or other cost function during training. Since the FOM is calculated by sorting true hits and false alarms separately for each target class and forming detection versus false alarm curves, these measures and their derivatives can not be computed analytically. Instead, the FOM and its derivative are computed using fast sort routines. These routines insert a new

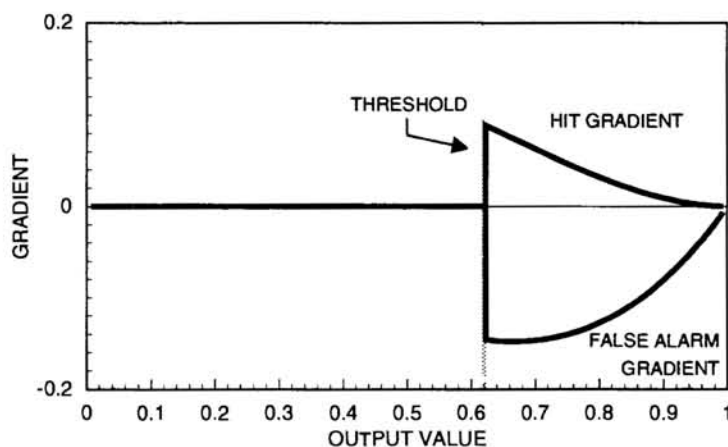

Figure 3. The gradient for a sigmoid output unit when the target value for true hits is set to 1.0 and the target value for false alarms is set to 0.0.

putative hit into an already sorted list and calculate the change in the FOM caused by that insertion. The running putative hit list used to compute the FOM is updated after every new putative hit is observed and it must contain all putative hits observed during the most recent past training cycle through all training patterns. The gradient estimate is smoothed over nearby putative hit scores to account for the quantized nature of detection versus false alarm rate curves.

Figure 4 shows plots of linearly scaled gradients for the 20-word hybrid wordspotter. Each value on the curve represents the smoothed change in the FOM that occurs when a single hit or false alarm with the specified normalized log output score is inserted into the current putative hit list. Gradients are positive for putative hits corresponding to true hits and negative for false alarms. They also fall off to zero for putative hits with extremely high or low scores. Shapes of these curves vary across words. The relative importance of a hit or false alarm, the normalized output score which results in high gradient values, and the shape of the gradient curve varies. Use of a squared error or other cost function with sigmoid output nodes would not generate this variety of gradients or automatically identify the range of putative hit scores where gradients should be high. Application of FOM training requires only the gradients shown in these curves with no supplementary thresholds. Patterns with low and high inputs will have a minimal effect during training without using thresholds because they produce gradients near zero.

Different keywords have dramatically different gradients. For example, *credit-card* is long and the detection rate is high. The overall FOM thus doesn't change much if more true hits are found. A high scoring false alarm, however, decreases the FOM drastically. There is thus a large negative gradient for false alarms for *credit-card*. The keywords *account* and *check* are usually short in duration and thus more difficult to detect, thus any increase in number of true hits strongly increases the overall FOM. On the other hand, since in this database, the words *account* and *check* occur much less frequently than *credit-card*, a high scoring false alarm for the words *account* and *check* has less impact on the overall FOM. The gradient for false alarms for these words is thus correspondingly smaller. Comparing the curves in Figure 3 with the fixed prototypical curve in Figure 4 demonstrates the dramatic differences in gradients that occur when the gradient is calculated to maximize the FOM directly instead of using a threshold with sigmoid output nodes.

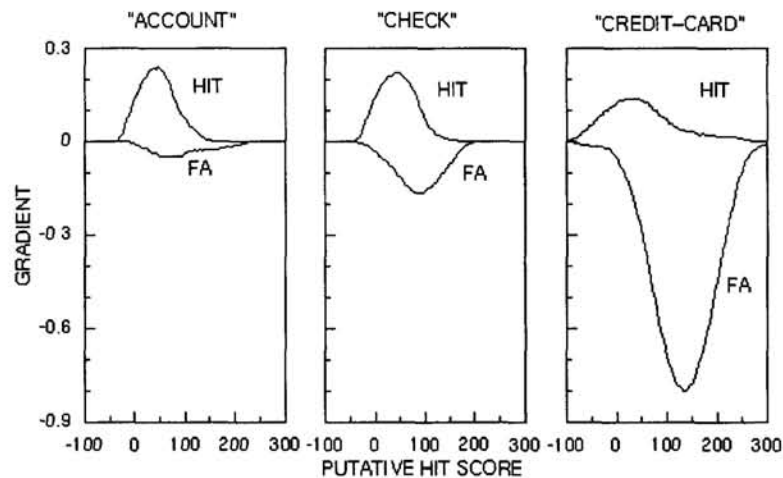

Figure 4.  Figure of merit gradients computed for true hits (HIT) and false alarms (FA) with scores ranging from -100 to 300 for the keywords *account*, *check*, and *credit-card*.

FOM training is a general technique that can applied to any "spotting" task where a set of targets must be discriminated from background inputs. FOM training was successfully tested using the hybrid radial basis function (RBF) – hidden Markov model (HMM) keyword spotter described in (Lippmann, 1993).

## 4   IMPLEMENTATION OF FOM TRAINING

FOM training is applied to our high-performance HMM wordspotter after forward-backward training is complete. Word models in the HMM wordspotter are first used to spot on training conversations. The FOM gradient of each putative hit is calculated when this hit is inserted into the putative hit list. The speech segment corresponding to a putative hit is excised from the conversation speech file and the corresponding keyword model is used to match each frame with a particular state in the model using a Viterbi backtrace (shown in Figure 5.) The gradient is then used to adjust the location of each Gaussian component in a node as in RBF classifiers (Lippmann, 1993) and also the *state weight* of each state. The *state weight* is a penalty added for each frame assigned to a state. The weight for each individual state is adjusted according to how important each state is to the detection of the keyword. For example, many false alarms for the word *card* are words that sound like part of the keyword such as *hard* or *far*. The first few states of the *card* model represent the sound /k/ and false alarms stay in these front states only a short time. If the state weight of the first few states of the *card* model is large, then a true hit has a larger score than false alarms.

The putative hit score which is used to detect peaks representing putative hits is generated according to

$$S_{total} = S_{keyword} - S_{filler}. \qquad \text{(EQ 1)}$$

In this equation, $S_{total}$ is the putative hit score, $S_{keyword}$ is the log Viterbi score in the

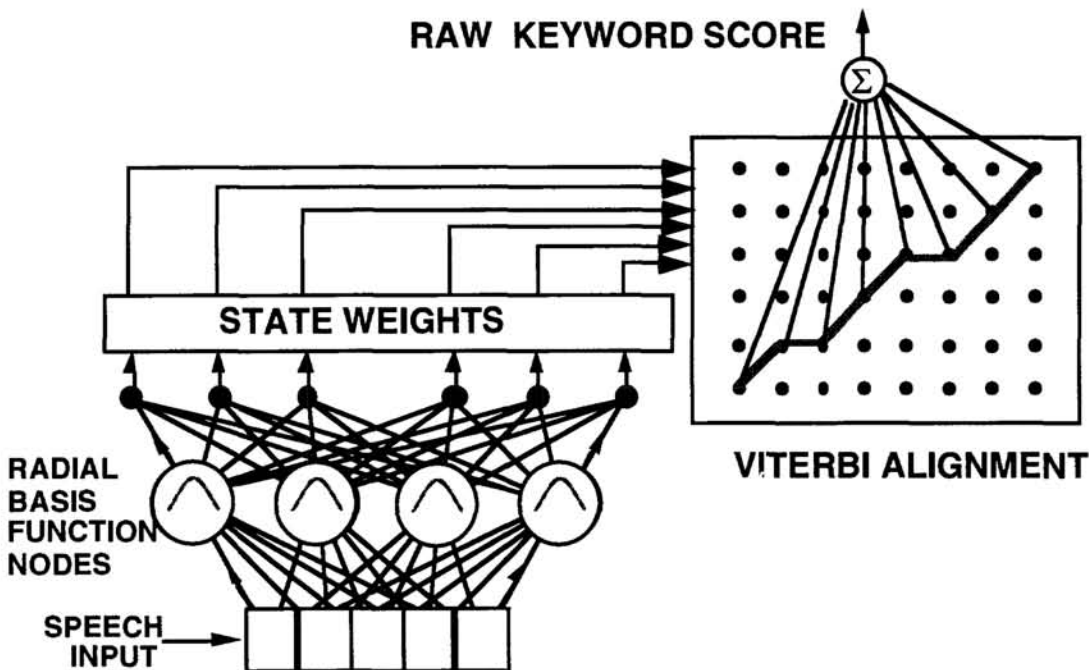

Figure 5.   State weights and center updates are applied to the state that is matched to each frame in a Viterbi backtrace.

last node of a specific keyword model computed using the Viterbi algorithm from the beginning of the conversation to the frame where the putative hit ended, and $S_{filler}$ is the log Viterbi score in the last node of the filler model. The filler score is used to normalize the keyword score and approximate a posterior probability. The keyword score is calculated using a modified form of the Viterbi algorithm

$$\alpha_i(t+1) = max(\alpha_i(t) + a_{i,i}, \alpha_{i-1}(t) + a_{i-1,i}) + d_i(t, \mathbf{x}) + w_i. \qquad \text{(EQ 2)}$$

This equation is identical to the normal Viterbi recursion for left-to-right linear word models after initialization, except the extra state score $w_i$ is added. In this equation, $\alpha_i(t)$ is the log Viterbi score in node $i$ at time $t$, $a_{i,j}$ is the log of the transition probability from node $i$ to node $j$, and $d_i(t, \mathbf{x})$ is the log likelihood distance score for node $i$ for the input feature vector $\mathbf{x}$ at time $t$.

Word scores are computed and a peak-picking algorithm looks for maxima above a low threshold. After a peak representing a putative hit is detected, frames of a putative hit are aligned with the states in the keyword model using the Viterbi backtrace and both the means of Gaussians in each state and state weights of the keyword model are modified. State weights are modified according to

$$w_i(t+1) = w_i(t) + gradient \times \eta_{state} \times duration. \qquad \text{(EQ 3)}$$

In this equation, $w_i(t)$ is the state weight in node $i$ at time $t$, $gradient$ is the FOM gradient for the putative hit, $\eta_{state}$ is the stepsize for state weight adaptation, and $duration$ is the number of frames aligned to node $i$. If a true hit occurs, and the gradient is positive, the state weight is increased in proportion to the number of frames assigned to a state. If a false alarm occurs, the state weight is reduced in proportion to the number of frames assigned to a state. The state weight will thus be strongly positive if there are many more frames for a true hit that for a false alarm. It will be strongly negative if there are more frames for a false alarm than for a true hit. High state weight values should thus improve discrimination between true hits and false alarms.

The center of the Gaussian components within each node, which are similar to Gaussians in radial basis function networks, are modified according to

$$m_{ij}(t+1) = m_{ij}(t) + gradient \times \eta_{center} \times \frac{x_j(t) - m_{ij}(t)}{\sigma_{ij}}. \qquad \text{(EQ 4)}$$

In this equation, $m_{ij}(t)$ is the $j$th component of the mean vector for a Gaussian hidden node in HMM state $i$ at time $t$, $gradient$ is the FOM gradient, $\eta_{center}$ is the stepsize for moving Gaussian centers, $x_j(t)$ is the value of the $j$th component of the input feature vector at time $t$, and $\sigma_{ij}$ is the standard deviation of the $j$th component of the Gaussian hidden node in HMM state $i$.

For each true hit, the centers of Gaussian hidden nodes in a state move toward the observation vectors of frames assigned to a particular state. For a false alarm, the centers move away from the observation vectors that are assigned to a particular state. Over time, the centers move closer to the true hit observation vectors and further away from false alarm observation vectors.

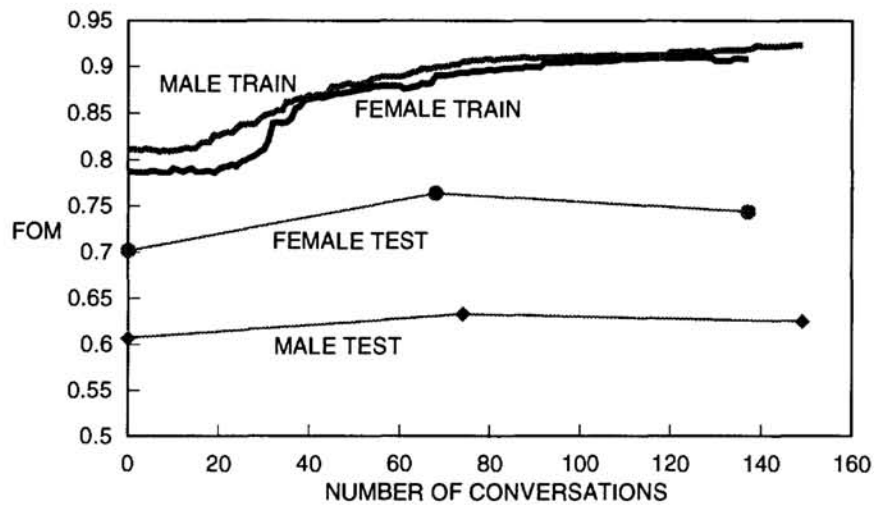

Figure 6. Change in FOM vs. the number of conversations that the models have been trained with. There were 25 male training conversations and 23 female training conversations.

## 5   EXPERIMENTAL RESULTS

Experiments were performed using a HMM wordspotter that was trained using maximum likelihood algorithm. More complicated models were created for words which occur frequently in the training set. The word models for *card* and *credit-card* were increased to four mixtures per state. The models for *cash, charge, check, credit, dollar, interest, money, month,* and *visa* were increased to two mixtures per state. All other word models had one mixture per state. The number of states per keyword is roughly 1.5 times the number of phonemes in each keyword. Covariance matrices were diagonal and variances were estimated separately for all states. All systems were trained on the first 50 talkers in the credit card training corpus and evaluated using the last 20 talkers.

An initial set of models was trained during 16 passes through the training data using whole-word training and Viterbi alignment on only the excised words from the training conversations. This training provided a FOM of 62.5% on the 20 evaluation talkers. Embedded forward-backward reestimation training was then performed where models of keywords and fillers are linked together and trained jointly on conversations which were split up into sentence-length fragments. This second stage of HMM training increased the FOM by two percentage points to 64.5%. The detection rate curves of these systems are shown in Figure 2.

FOM training was then performed for six passes through the training data. On each pass, conversations were presented in a new random order. The change in FOM for the training set and the evaluation set is shown in Figure 6. The FOM on the training data for both male and female talkers increased by more than 10 percentage points after roughly 50 conversations had been presented. The FOM on the evaluation data increased by 5.2 percentage points to 69.7% after three passes through the training data, but then decreased with further training. This result suggests that the extra structure learned during the final three training passes is overfitting the training data and providing poor performance on the evaluation set. Figure 7 shows the spectrograms of high scoring true hits and false alarms for the word *card* generated by our wordspotter. All false alarms shown are actually the occurrences of the word *car*. The spectrograms of the true hits and the false alarms are very similar and the actual excised speech segments are difficult even for humans to distinguish.

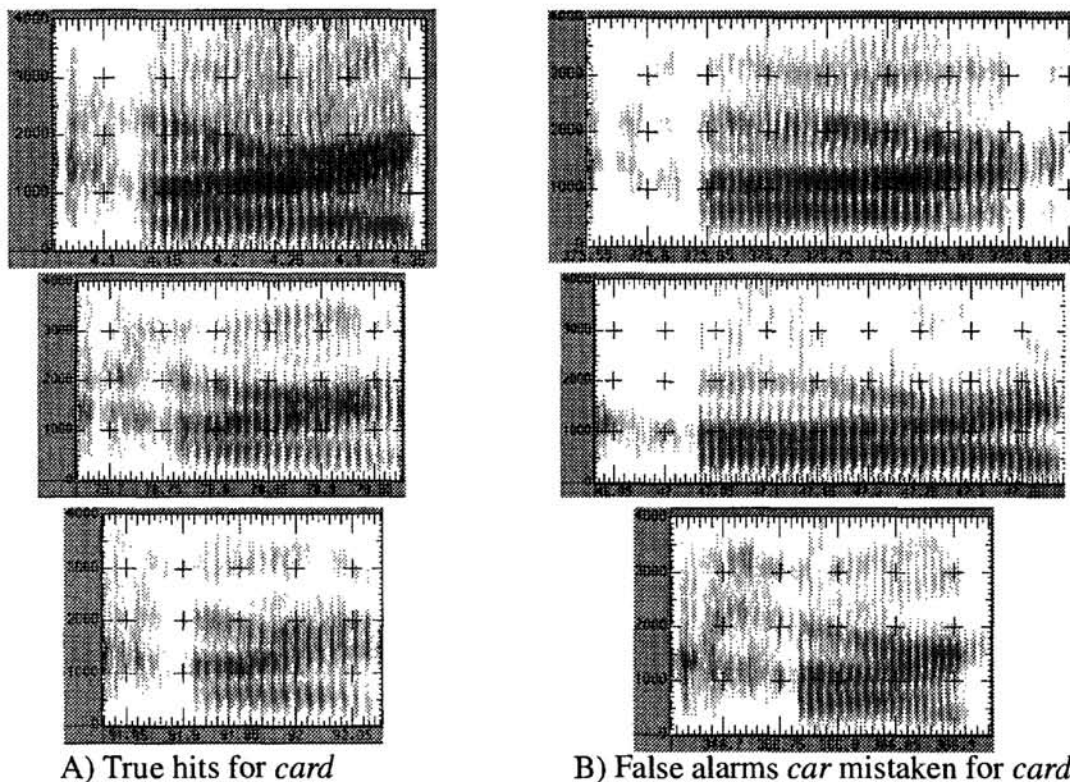

A) True hits for *card*        B) False alarms *car* mistaken for *card*

Figure 7. Spectrograms of high scoring true hit and false alarm for the word *card*.

## 6 SUMMARY

Detection of target signals embedded in a noisy background is a common and difficult problem distinct from the task of classification. The evaluation metric of a spotting system, called Figure of Merit (FOM), is also different from the classification accuracy used to evaluate classification systems. FOM training uses a gradient which directly reflects a putative hit's impact on the FOM to modify the parameters of the spotting system. FOM training does not require careful adjustment of thresholds and target values and has been applied to improve a wordspotter's FOM from 64.5% to 69.7% on the credit card database. FOM training can also be applied to other spotting tasks such as arrhythmia detection and address block location.

## ACKNOWLEDGEMENT

This work was sponsored by the Advanced Research Projects Agency. The views expressed are those of the authors and do not reflect the official policy or position of the U.S. Government. Portions of this work used the HTK Toolkit developed by Dr. Steve Young of Cambridge University.

## BIBLIOGRAPHY

R. Lippmann & E. Singer. (1993) Hybrid HMM/Neural-Network Approaches to Wordspotting. In *ICASSP '93*, volume I, pages 565-568.

J. Rohlicek et. al. (1993) Phonetic and Language Modeling for Wordspotting. In *ICASSP '93*, volume II, pages 459-462.

T. Zeppenfeld, R. Houghton & A. Waibel. (1993) Improving the MS-TDNN for Word Spotting. In *ICASSP '93*, volume II, pages 475-478.